# A Practice Strategy for Robot Learning Control

**Terence D. Sanger**
Department of Electrical Engineering and Computer Science
Massachusetts Institute of Technology, room E25-534
Cambridge, MA 02139
tds@ai.mit.edu

## Abstract

"Trajectory Extension Learning" is a new technique for Learning Control in Robots which assumes that there exists some parameter of the desired trajectory that can be smoothly varied from a region of easy solvability of the dynamics to a region of desired behavior which may have more difficult dynamics. By gradually varying the parameter, practice movements remain near the desired path while a Neural Network learns to approximate the inverse dynamics. For example, the average speed of motion might be varied, and the inverse dynamics can be "bootstrapped" from slow movements with simpler dynamics to fast movements. This provides an example of the more general concept of a "Practice Strategy" in which a sequence of intermediate tasks is used to simplify learning a complex task. I show an example of the application of this idea to a real 2-joint direct drive robot arm.

## 1 INTRODUCTION

The most general definition of Adaptive Control is one which includes any controller whose behavior changes in response to the controlled system's behavior. In practice, this definition is usually restricted to modifying a small number of controller parameters in order to maintain system stability or global asymptotic stability of the errors during execution of a single trajectory (Sastry and Bodson 1989, for review). Learning Control represents a second level of operation, since it uses Adaptive Con-

trol to modify parameters during repeated performance trials of a desired trajectory so that future trials result in greater accuracy (Arimoto *et al.* 1984). In this paper I present a third level called a "Practice Strategy", in which Learning Control is applied to a sequence of intermediate trajectories leading ultimately to the true desired trajectory. I claim that this can significantly increase learning speed and make learning possible for systems which would otherwise become unstable.

## 1.1    LEARNING CONTROL

During repeated practice of a single desired trajectory, the actual trajectory followed by the robot may be significantly different. Many Learning Control algorithms modify the commands stored in a sequence memory to minimize this difference (Atkeson 1989, for review). However, the performance errors are usually measured in a sensory coordinate system, while command corrections must be made in the motor coordinate system. If the relationship between these two coordinate systems is not known, then command corrections might be in the wrong direction and inadvertently worsen performance. However, if the practice trajectory is close to the desired trajectory, then the errors will be small and the relationship between command and sensory errors can be approximated by the system Jacobian.

An alternative to a stored command sequence is to use a Neural Network to learn an approximation to the inverse dynamics in the region of interest (Sanner and Slotine 1992, Yabuta and Yamada 1991, Atkeson 1989). In this case, the commands and results from the actual movement are used as training data for the network, and smoothness properties are assumed such that the error on the desired trajectory will decrease. However, a significant problem with this method is that if the actual practice trajectory is far from the desired trajectory, then its inverse dynamics information will be of little use in training the inverse dynamics for the desired trajectory. In fact, the network may achieve perfect approximation on the actual trajectory while still making significant errors on the desired trajectory. In this case, learning will stop (since the training error is zero) leading to the phenomenon of "learning lock-up" (An *et al.* 1988). So whether Learning Control uses a sequence memory or a Neural Network, learning may proceed poorly if large errors are made during the initial practice movements.

## 1.2    PRACTICE STRATEGIES

I define a "practice strategy" as a sequence of trajectories such that the first element in the sequence is any previously learned trajectory, and the last element in the sequence is the ultimate desired trajectory. A well designed practice strategy will result in a seqence for which learning control of the trajectory for any particular step is simplified if prior steps have already been learned. This will occur if learning of prior trajectories reduces the initial performance error for subsequent trajectories, so that a network will be less likely to experience learning lock-up.

One example of a practice strategy is a three-step sequence in which the intermediate step is a set of independently executable subtasks which partition the desired trajectory into discrete pieces. Another example is a multi-step sequence in which intermediate steps are a set of trajectories which are somehow related to the desired trajectory. In this paper I present a multi-step sequence which gradually

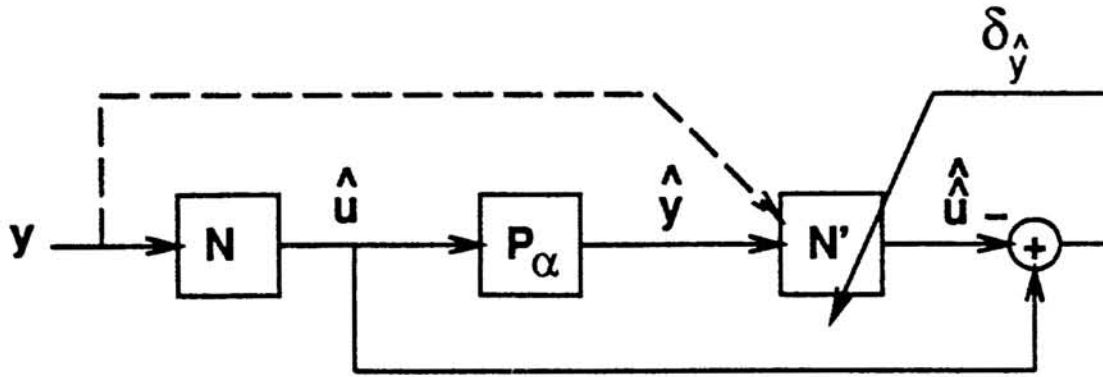

Figure 1: Training signals for network learning.

transforms some known trajectory into the desired trajectory by varying a single parameter. This method has the advantage of not requiring detailed knowledge of the task structure in order to break it up into meaningful subtasks, and conditions for convergence can be stated explicitly. It has a close relationship to Continuation Methods for solving differential equations, and can be considered to be a particular application of the Banach Extension Theorem.

## 2   METHODS

As in (Sanger 1992), we need to specify 4 aspects of the use of a neural network within a control system:

1. the networks' function in the control system,

2. the network learning algorithm which modifies the connection weights,

3. the training signals used for network learning, and

4. the practice strategy used to generate sample movements.

The network's function is to learn the inverse dynamics of an equilibrium-point controlled plant (Shadmehr 1990). The LMS-tree learning algorithm trains the network (Sanger 1991b, Sanger 1991a). The training signals are determined from the actual practice data using either "Actual Trajectory Training" or "Desired Trajectory Training", as defined below. And the practice strategy is "Trajectory Extension Learning", in which a parameter of the movement is gradually modified during training.

## 2.1  TRAINING SIGNALS

Figure 1 shows the general structure of the network and training signals. A desired trajectory $y$ is fed into the network $N$ to yield an estimated command $\hat{u}$. This command is then applied to the plant $P_\alpha$ where the subscript indicates that the plant is parameterized by the variable $\alpha$. Although the true command $u$ which achieves $y$ is unknown, we do know that the estimated command $\hat{u}$ produces $\hat{y}$, so these signals are used for training by comparing the network response to $\hat{y}$ given by $\hat{\hat{u}} = N\hat{y}$ to the known value $\hat{u}$ and subtracting these to yield the training error $\delta_{\hat{y}}$.

Normally, network training would use this error signal to modify the network output for inputs near $\hat{y}$, and I refer to this as "Actual Trajectory Training". However, if $\hat{y}$ is far from $y$ then no change in response may occur at $y$ and this may lead even more quickly to learning lock-up. Therefore an alternative is to use the error $\delta_{\hat{y}}$ to train the network output for inputs near $y$. I refer to this as "Desired Trajectory Training", and in the figure it is represented by the dotted arrow.

The following discussion will summarize the convergence conditions and theorems presented in (Sanger 1992).

Define
$$Ru \doteq (I - NP(x))u = u - \hat{u}$$
to be an operator which maps commands into command errors for states $x$ on the desired trajectory. Similarly, let
$$\hat{R}\hat{u} = (I - NP(\hat{x}))\hat{u} = \hat{u} - \hat{\hat{u}}$$
map commands into command errors for states $\hat{x}$ on the actual trajectory.

Convergence depends upon the following assumptions:

**A1:** The plant $P$ is smooth and invertible with respect to both the state $x$ and the input $u$ with Lipschitz constants $k_x$ and $k_u$, and it has stable zero-dynamics.

**A2:** The network $N$ is smooth with Lipschitz constant $k_N$.

**A3:** Network learning reduces the error in response to a pair $(y, \delta_y)$.

**A4:** The change in network output in response to training is smooth with Lipschitz constant $k_L$.

**A5:** There exists a smoothly controllable parameter $\alpha$ such that an inverse dynamics solution is available at $\alpha = \alpha_0$, and the desired performance occurs when $\alpha = \alpha_d$.

**A6:** The change in command required to produce a desired output after any change in $\alpha$ is bounded by the change in $\alpha$ multiplied by a constant $k_\alpha$.

**A7:** The change in plant response for any fixed input is bounded by the change in $\alpha$ multiplied by a constant $k_p$.

Under assumptions A1-A3 we can prove convergence of Desired Trajectory Training:

**Theorem 1:**
*If there exists a $k_{R_n}$ such that*
$$\|R_n u - \hat{R}_n \hat{u}\| < k_{R_n}\|u - \hat{u}\|$$

*then if the learning rate* $0 < \gamma \leq 1$,

$$\|R_{n+1}u\| < (1 - \gamma + \gamma k_{R_n}))\|R_n u\|.$$

If $k_{R_n} < 1$ and $\gamma \leq 1$, then the network output $\hat{u}$ approaches the correct command $u$.

Under assumptions A1-A4, we can prove convergence of Actual Trajectory Training:

**Theorem 2:**
*If there exists a* $k_{R_n}$ *such that*

$$\|R_n u - \hat{R}_n \hat{u}\| < k_{R_n}\|u - \hat{u}\|$$

*then if the learning rate* $0 < \gamma \leq 1$,

$$\|R_{n+1}u\| < (1 - \gamma + \gamma k_L k_u + \gamma k_{R_n})\|R_n u\| + \gamma k_L k_x\|x - \hat{x}\|$$

## 2.2   TRAJECTORY EXTENSION LEARNING

Let $\alpha$ be some modifiable parameter of the plant such that for $\alpha = \alpha_0$ there exists a simple inverse dynamics solution, and we seek a solution when $\alpha = \alpha_d$. For example, if the plant uses Equilibrium Point Control (Shadmehr 1990), then at low speeds the inverse dynamics behave like a perfect servo controller yielding desired trajectories without the need to solve the dynamics. We can continue to train a learning controller as the average speed of movement ($\alpha$) is gradually increased. The inverse dynamics learned at one speed provide an approximation to the inverse dynamics for a slightly faster speed, and thus the performance errors remain small during practice. This leads to significantly faster learning rates and greater likelihood that the conditions for convergence at any given speed will be satisfied. Note that unlike traditional learning schemes, the error does not decrease monotonically with practice, but instead maintains a steady magnitude as the speed increases, until the network is no longer able to approximate the inverse dynamics.

The following is a summary of a result from (Sanger 1992). Let $\alpha$ change from $\alpha_1$ to $\alpha_2$, and let $P = P_{\alpha_1}$ and $P' = P_{\alpha_2}$. Then under assumptions A1-A7 we can prove convergence of Trajectory Extension Learning:

**Theorem 3:**
*If there exists a* $k_R$ *such that for* $\alpha = \alpha_1$

$$\|Ru - \hat{R}\hat{u}\| < k_R\|u - \hat{u}\|$$

*then for* $\alpha = \alpha_2$

$$\|R'u' - \hat{R}'\hat{u}\| < k_R\|u' - \hat{u}\| + (2k_\alpha + k_N k_p)|\alpha_2 - \alpha_1|$$

This shows that given the smoothness assumptions and a small enough change in $\alpha$, the error will continue to decrease.

## 3  EXAMPLE

Figure 2 shows the result of 15 learning trials performed by a real direct-drive two-joint robot arm on a sampled desired trajectory. The initial trial required 11.5 seconds to execute, and the speed was gradually increased until the final trial required only 4.5 seconds. Simulated equilibrium point control was used (Bizzi *et al.* 1984) with stiffness and damping coefficients of 15 nm/rad and 1.5 nm/rad/sec, respectively. The grey line in figure 2 shows the equilibrium point control signal which generated the actual movement represented by the solid line. The difference between these two indicates the nontrivial nature of the dynamics calculations required to derive the control signal from the desired trajectory. Note that without Trajectory Extension Learning, the network does not converge and the arm becomes unstable. The neural network was an LMS tree (Sanger 1991b, Sanger 1991a) with 10 Gaussian basis functions for each of the 6 input dimensions, and a total of 15 subtrees were grown per joint (see (Sanger 1992) for further explanation).

## 4  CONCLUSION

Trajectory Extension Learning is one example of the way in which a practice strategy can be used to improve convergence for Learning Control. This or other types of practice strategies might be able to increase the performance of many different types of learning algorithms both within and outside the Control domain. Such strategies may also provide a theoretical model for the practice strategies used by humans to learn complex tasks, and the theoretical analysis and convergence conditions could potentially lead to a deeper understanding of human motor learning and successful techniques for optimizing performance.

### Acknowledgements

Thanks are due to Simon Giszter, Reza Shadmehr, Sandro Mussa-Ivaldi, Emilio Bizzi, and many people at the NIPS conference for their comments and criticisms. This report describes research done within the laboratory of Dr. Emilio Bizzi in the department of Brain and Cognitive Sciences at MIT. The author was supported during this work by a National Defense Science and Engineering Graduate Fellowship, and by NIH grants 5R37AR26710 and 5R01NS09343 to Dr. Bizzi.

### References

An C. H., Atkeson C. G., Hollerbach J. M., 1988, *Model-Based Control of a Robot Manipulator*, MIT Press, Cambridge, MA.

Arimoto S., Kawamura S., Miyazaki F., 1984, Bettering operation of robots by learning, *Journal of Robotic Systems*, 1(2):123–140.

Atkeson C. G., 1989, Learning arm kinematics and dynamics, *Ann. Rev. Neurosci.*, 12:157–183.

Bizzi E., Accornero N., Chapple W., Hogan N., 1984, Posture control and trajectory formation during arm movement, *J. Neurosci*, 4:2738–2744.

Sanger T. D., 1991a, A tree-structured adaptive network for function approximation in high dimensional spaces, *IEEE Trans. Neural Networks*, 2(2):285–293.

Sanger T. D., 1991b, A tree-structured algorithm for reducing computation in networks with separable basis functions, *Neural Computation*, 3(1):67–78.

Sanger T. D., 1992, Neural network learning control of robot manipulators using gradually increasing task difficulty, submitted to *IEEE Trans. Robotics and Automation*.

Sanner R. M., Slotine J.-J. E., 1992, Gaussian networks for direct adaptive control, IEEE Trans. Neural Networks, in press. Also MIT NSL Report 910303, 910503, March 1991 and Proc. American Control Conference, Boston pages 2153–2159, June 1991.

Sastry S., Bodson M., 1989, *Adaptive Control: Stability, Convergence, and Robustness*, Prentice Hall, New Jersey.

Shadmehr R., 1990, Learning virtual equilibrium trajectories for control of a robot arm, *Neural Computation*, 2:436–446.

Yabuta T., Yamada T., 1991, Learning control using neural networks, *Proc. IEEE Int'l Conf. on Robotics and Automation, Sacramento*, pages 740–745.

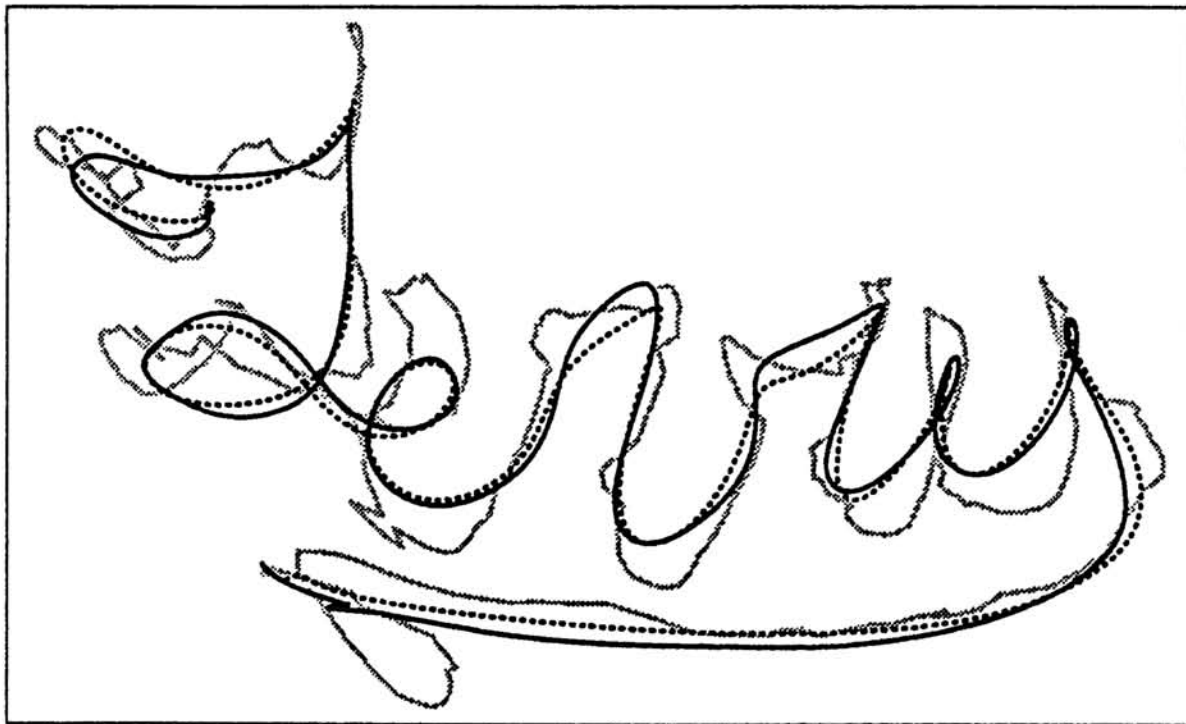

Figure 2: Dotted line is the desired trajectory, solid line is the actual trajectory, and the grey line is the equilibrium point control trajectory.